# PAC-Bayes Bounds for the Risk of the Majority Vote and the Variance of the Gibbs Classifier

**Alexandre Lacasse, François Laviolette and Mario Marchand**
Département IFT-GLO
Université Laval
Québec, Canada
`Firstname.Secondname@ift.ulaval.ca`

**Pascal Germain**
Département IFT-GLO
Université Laval Québec, Canada
`Pascal.Germain.1@ulaval.ca`

**Nicolas Usunier**
Laboratoire d'informatique de Paris 6
Université Pierre et Marie Curie, Paris, France
`Nicolas.Usunier@lip6.fr`

## Abstract

We propose new PAC-Bayes bounds for the risk of the weighted majority vote that depend on the mean and variance of the error of its associated Gibbs classifier. We show that these bounds can be smaller than the risk of the Gibbs classifier and can be arbitrarily close to zero even if the risk of the Gibbs classifier is close to 1/2. Moreover, we show that these bounds can be uniformly estimated on the training data for all possible posteriors $Q$. Moreover, they can be improved by using a large sample of unlabelled data.

## 1 Introduction

The PAC-Bayes approach, initiated by [1], aims at providing PAC guarantees to "Bayesian-like" learning algorithms. Within this approach, we consider a *prior*[1] *distribution $P$* over a space of classifiers that characterizes our prior belief about good classifiers (before the observation of the data) and a *posterior distribution $Q$* (over the same space of classifiers) that takes into account the additional information provided by the training data. A remarkable result, known as the "PAC-Bayes theorem", provides a risk bound for the $Q$-weigthed majority-vote by bounding the risk of an associated stochastic classifier called the *Gibbs classifier*. Bounds previously existed which showed that you can de-randomize back to the Majority Vote classifier, but these come at a cost of worse risk. Naively, one would expect that the de-randomized classifier would perform better. Indeed, it is well-known that voting can dramatically improve performance when the "community" of classifiers tend to compensate the individual errors. The actual PAC-Bayes framework is currently unable to evaluate whether or not this compensation occurs. Consequently, this framework can not currently help in producing highly accurate voted combinations of classifiers.

In this paper, we present new PAC-Bayes bounds on the risk of the Majority Vote classifier based on the estimation of the mean and *variance* of the errors of the associated Gibbs classifier. These bounds allow to prove that a sufficient condition to provide an accurate combination is (1) that the error of the Gibbs classifier is less than half and (2) the mean pairwise covariance of the errors of the classifiers appearing in the vote is small. In general, the bound allows to detect when the voted combination provably outperforms its associated Gibbs classifier.

## 2  Basic Definitions

We consider binary classification problems where the input space $\mathcal{X}$ consists of an arbitrary subset of $\mathbb{R}^n$ and the output space $\mathcal{Y} = \{-1, +1\}$. An example $\mathbf{z} \stackrel{\text{def}}{=} (\mathbf{x}, y)$ is an input-output pair where $\mathbf{x} \in \mathcal{X}$ and $y \in \mathcal{Y}$. Throughout the paper, we adopt the PAC setting where each example $\mathbf{z}$ is drawn according to a fixed, but unknown, probability distribution $D$ on $\mathcal{X} \times \mathcal{Y}$.

We consider learning algorithms that work in a fixed hypothesis space $\mathcal{H}$ of binary classifiers (defined without reference to the training data). The *risk* $R(h)$ of any classifier $h\colon \mathcal{X} \to \mathcal{Y}$ is defined as the probability that $h$ misclassifies an example drawn according to $D$:

$$R(h) \stackrel{\text{def}}{=} \Pr_{(\mathbf{x}, y) \sim D} \left( h(\mathbf{x}) \neq y \right) = \mathop{\mathbf{E}}_{(\mathbf{x}, y) \sim D} I(h(\mathbf{x}) \neq y),$$

where $I(a) = 1$ if predicate $a$ is true and $0$ otherwise.

Given a training set $S$, $m$ will always represent its number of examples. Moreover, if $S = \langle \mathbf{z}_1, \ldots, \mathbf{z}_m \rangle$, the *empirical risk* $R_S(h)$ on $S$, of any classifier $h$, is defined according to:

$$R_S(h) \stackrel{\text{def}}{=} \frac{1}{m} \sum_{i=1}^{m} I(h(\mathbf{x}_i) \neq y_i).$$

After observing the training set $S$, the task of the learner is to choose a *posterior* distribution $Q$ over $\mathcal{H}$ such that the $Q$-weighted Majority Vote classifier, $B_Q$, will have the smallest possible risk. On any input training example $\mathbf{x}$, the output, $B_Q(\mathbf{x})$, of the Majority Vote classifier $B_Q$ (also called the Bayes classifier) is given by:

$$B_Q(\mathbf{x}) \stackrel{\text{def}}{=} \operatorname{sgn} \left[ \mathop{\mathbf{E}}_{h \sim Q} h(\mathbf{x}) \right],$$

where $\operatorname{sgn}(s) = +1$ if real number $s > 0$ and $\operatorname{sgn}(s) = -1$ otherwise. The output of the deterministic Majority Vote classifier $B_Q$ is thus closely related to the output of a stochastic classifier called the *Gibbs* classifier. To classify an input example $\mathbf{x}$, the Gibbs classifier $G_Q$ chooses randomly a (deterministic) classifier $h$ according to $Q$ to classify $\mathbf{x}$. The true risk $R(G_Q)$ and the empirical risk $R_S(G_Q)$ of the Gibbs classifier are thus given by:

$$R(G_Q) \stackrel{\text{def}}{=} \mathop{\mathbf{E}}_{h \sim Q} R(h) = \mathop{\mathbf{E}}_{h \sim Q} \mathop{\mathbf{E}}_{(\mathbf{x}, y) \sim D} I(h(\mathbf{x}) \neq y) \tag{1}$$

$$R_S(G_Q) \stackrel{\text{def}}{=} \mathop{\mathbf{E}}_{h \sim Q} R_S(h) = \mathop{\mathbf{E}}_{h \sim Q} \frac{1}{m} \sum_{i=1}^{m} I(h(\mathbf{x}_i) \neq y_i). \tag{2}$$

The PAC-Bayes theorem gives a tight risk bound for the Gibbs classifier $G_Q$ that depends on how far is the chosen posterior $Q$ from a *prior* $P$ that must be chosen before observing the data. The PAC-Bayes theorem was first proposed by [2]. The bound presented here can be found in [3].

**Theorem 1** *(PAC-Bayes Theorem) For any prior distribution $P$ over $\mathcal{H}$, and any $\delta \in \, ]0, 1]$, we have*

$$\Pr_{S \sim D^m} \left( \forall Q \text{ over } \mathcal{H}: \ \operatorname{kl}(R_S(G_Q) \| R(G_Q)) \leq \frac{1}{m} \left[ \operatorname{KL}(Q\|P) + \ln \frac{m+1}{\delta} \right] \right) \geq 1 - \delta \,,$$

*where $\operatorname{KL}(Q\|P)$ is the Kullback-Leibler divergence between $Q$ and $P$:*

$$\operatorname{KL}(Q\|P) \stackrel{\text{def}}{=} \mathop{\mathbf{E}}_{h \sim Q} \ln \frac{Q(h)}{P(h)} \,,$$

*and where $\operatorname{kl}(q\|p)$ is the Kullback-Leibler divergence between the Bernoulli distributions with probability of success $q$ and probability of success $p$: $\operatorname{kl}(q\|p) \stackrel{\text{def}}{=} q \ln \frac{q}{p} + (1-q) \ln \frac{1-q}{1-p}$.*

This theorem has recently been generalized by [4] to the sample-compression setting. In this paper, however, we restrict ourselves to the more common case where the set $\mathcal{H}$ of classifiers is defined without reference to the training data.

A bound given for the risk of Gibbs classifiers can straightforwardly be turned into a bound for the risk of Majority Vote classifiers. Indeed, whenever $B_Q$ misclassifies $\mathbf{x}$, at least half of the classifiers (under measure $Q$), misclassifies $\mathbf{x}$. It follows that the error rate of $G_Q$ is at least half of the error rate of $B_Q$. Hence $R(B_Q) \leq 2R(G_Q)$. A method to decrease the $R(B_Q)/R(G_Q)$ ratio to $1 + \epsilon$ (for some small positive $\epsilon$) has been proposed by [5] for large-margin classifiers. For a suitably chosen prior and posterior, [5] have also shown that $R_S(G_Q)$ is small when the corresponding Majority Vote classifier $B_Q$ achieves a large separating margin on the training data. Consequently, the PAC-Bayes theorem yields a tight risk bound for large margin classifiers.

Even if we can imagine situations where $R(B_Q) > R(G_Q)$, they have been rarely encountered in practice. In fact, situations where $R(B_Q)$ is much smaller than $R(G_Q)$ seem to occur much more often. For example, consider the extreme case where the true label $y$ of $\mathbf{x}$ is 1 iff $\mathbf{E}_{h \sim Q} h(\mathbf{x}) > 1/2$. In this case $R(B_Q) = 0$ whereas $R(G_Q)$ can be as high as $1/2 - \epsilon$ for some arbitrary small $\epsilon$. The situations where $R(B_Q)$ is much smaller than $R(G_Q)$ are not captured by the PAC-Bayes theorem. In the next section, we provide a bound on $R(B_Q)$ that depends on $R(G_Q)$ *and* other properties that can be estimated from the training data. This bound can be arbitrary close to 0 even for a large $R(G_Q)$ as long as $R(G_Q) < 1/2$ and as long as we have a sufficiently large population of classifiers for which their errors are sufficiently "uncorrelated".

## 3   A Bound on $R(B_Q)$ that Can Be Much Smaller than $R(G_Q)$

All of our relations between $R(B_Q)$ and $R(G_Q)$ arise by considering the $Q$-weight $W_Q(\mathbf{x}, y)$ of classifiers making errors on example $(\mathbf{x}, y)$:

$$W_Q(\mathbf{x}, y) \overset{\text{def}}{=} \underset{h \sim Q}{\mathbf{E}} \, I(h(\mathbf{x}) \neq y) \,. \tag{3}$$

Clearly, we have:   $\underset{(\mathbf{x},y) \sim D}{\Pr} (W_Q(\mathbf{x}, y) > 1/2) \;\leq\; R(B_Q) \;\leq\; \underset{(\mathbf{x},y) \sim D}{\Pr} (W_Q(\mathbf{x}, y) \geq 1/2).$ $\tag{4}$

Hence, $\underset{(\mathbf{x},y) \sim D}{\Pr} (W_Q(\mathbf{x}, y) \geq 1/2)$ gives a very tight upper bound on $R(B_Q)$. Moreover,

$$\underset{(\mathbf{x},y) \sim D}{\mathbf{E}} W_Q(\mathbf{x}, y) \;=\; \underset{(\mathbf{x},y) \sim D}{\mathbf{E}} \underset{h \sim Q}{\mathbf{E}} I(h(\mathbf{x}) \neq y) \;=\; R(G_Q) \tag{5}$$

and

$$
\begin{aligned}
\underset{(\mathbf{x},y) \sim D}{\mathbf{Var}} (W_Q) \;&=\; \underset{(\mathbf{x},y) \sim D}{\mathbf{E}} \left( (W_Q)^2 - \left( \underset{(\mathbf{x},y) \sim D}{\mathbf{E}} W_Q \right)^2 \right) \\
&=\; \underset{(\mathbf{x},y) \sim D}{\mathbf{E}} \left( \underset{h_1 \sim Q}{\mathbf{E}} I(h_1(\mathbf{x}) \neq y) \underset{h_2 \sim Q}{\mathbf{E}} I(h_2(\mathbf{x}) \neq y) \right) - R^2(G_Q) \\
&=\; \underset{h_1 \sim Q}{\mathbf{E}} \underset{h_2 \sim Q}{\mathbf{E}} \left( \underset{(\mathbf{x},y) \sim D}{\mathbf{E}} I(h_1(\mathbf{x}) \neq y) I(h_2(\mathbf{x}) \neq y) - R(h_1) R(h_2) \right) \\
&\overset{\text{def}}{=}\; \underset{h_1 \sim Q}{\mathbf{E}} \underset{h_2 \sim Q}{\mathbf{E}} \mathrm{cov}_{\mathrm{err}}(h_1, h_2) \,,
\end{aligned}
\tag{6}
$$

where $\mathrm{cov}_{\mathrm{err}}(h_1, h_2)$ denotes the *covariance of the errors of $h_1$ and $h_2$ on examples drawn by $D$.*

The next theorem is therefore a direct consequence of the *one-sided* Chebychev (or Cantelli-Chebychev) inequality [6]:   $\Pr\left(W_Q \geq a + \mathbf{E}(W_Q)\right) \;\leq\; \dfrac{\mathbf{Var}(W_Q)}{\mathbf{Var}(W_Q) + a^2}$   for any $a > 0$.

**Theorem 2** *For any distribution $Q$ over a class of classifiers, if $R(G_Q) \leq 1/2$ then we have*

$$R(B_Q) \leq \dfrac{\underset{(\mathbf{x},y) \sim D}{\mathbf{Var}} (W_Q)}{\underset{(\mathbf{x},y) \sim D}{\mathbf{Var}} (W_Q) + (1/2 - R(G_Q))^2} = \dfrac{\underset{(\mathbf{x},y) \sim D}{\mathbf{Var}} (1 - 2W_Q)}{\underset{(\mathbf{x},y) \sim D}{\mathbf{E}} (1 - 2W_Q)^2} \overset{\text{def}}{=} C_Q.$$

We will always use here the first form of $C_Q$. However, note that $1 - 2W_Q = \sum_{h \in \mathcal{H}} Q(h) y h(\mathbf{x})$ is just the *margin* of the $Q$-convex combination realized on $(\mathbf{x}, y)$. Hence, the second form of $C_Q$ is simply the variance of the margin divided by its second moment!

The looser *two-sided* Chebychev inequality was used in [7] to bound the risk of random forests. However, the one-sided bound $C_Q$ is much tighter. For example, the two-sided bound in [7] diverges when $R(G_Q) \to 1/2$, but $C_Q \leq 1$ whenever $R(G_Q) \leq 1/2$. In fact, as explained in [8], the one-sided Chebychev bound is the tightest possible upper bound for any random variable which is based only on its expectation and variance.

The next result shows that, when the number of voters tends to infinity (and the weight of each voter tends to zero), the variance of $W_Q$ will tend to 0 provided that the average of the covariance of the risks of all pairs of distinct voters is $\leq 0$. In particular, the variance will always tend to 0 if the risk of the voters are pairwise independent.

**Proposition 3** *For any countable class $\mathcal{H}$ of classifiers and any distribution $Q$ over $\mathcal{H}$, we have*

$$\operatorname*{\mathbf{Var}}_{(\mathbf{x},y)\sim D}(W_Q) \leq \frac{1}{4}\sum_{h\in\mathcal{H}}Q^2(h) + \sum_{h_1\in\mathcal{H}}\sum_{\substack{h_2\in\mathcal{H}:\\h_2\neq h_1}}Q(h_1)Q(h_2)\operatorname{cov_{err}}(h_1,h_2).$$

The proof is straightforward and is left to the reader. The key observation that comes out of this result is that $\sum_{h\in\mathcal{H}}Q^2(h)$ is usually much smaller than one. Consider, for example, the case where $Q$ is uniform on $\mathcal{H}$ with $|\mathcal{H}| = n$. Then $q = \sum_{h\in\mathcal{H}}Q^2(h) = 1/n$. Moreover, if $\operatorname{cov_{err}}(h_1,h_2) \leq 0$ for each pair of distinct classifiers in $\mathcal{H}$, then $\mathbf{Var}(W_Q) \leq 1/(4n)$. Hence, in these cases, we have that $C_Q \in \mathcal{O}(1/n)$ whenever $1/2 - R(G_Q)$ is larger than some positive constant independent of $n$. Thus, even when $R(G_Q)$ is large, we see that $R(B_Q)$ can be arbitrarily close to 0 as we increase the number of classifiers having non-positive pairwise covariance of their risk.

To further motivate the use of $C_Q$, we have investigated, on several UCI binary classification data sets, how $R(G_Q)$, $\mathbf{Var}(W_Q)$ and $C_Q$ are respectively related to $R(B_Q)$. The results of Figure 1 have been obtained with the Adaboost [9] algorithm used with "decision stumps" as weak learners. Each data set was split in two halves: one used for training and the other for testing. In the chart relating $R(G_Q)$ and $R(B_Q)$, we see that we almost always have $R(B_Q) < R(G_Q)$. There is, however, no clear correlation between $R(B_Q)$ and $R(G_Q)$. We also see no clear correlation between $R(B_Q)$ and $\mathbf{Var}(W_Q)$ in the second chart. In contrast, the chart of $C_Q$ vs $R(B_Q)$ shows a strong correlation. Indeed, it is almost a linear relation!

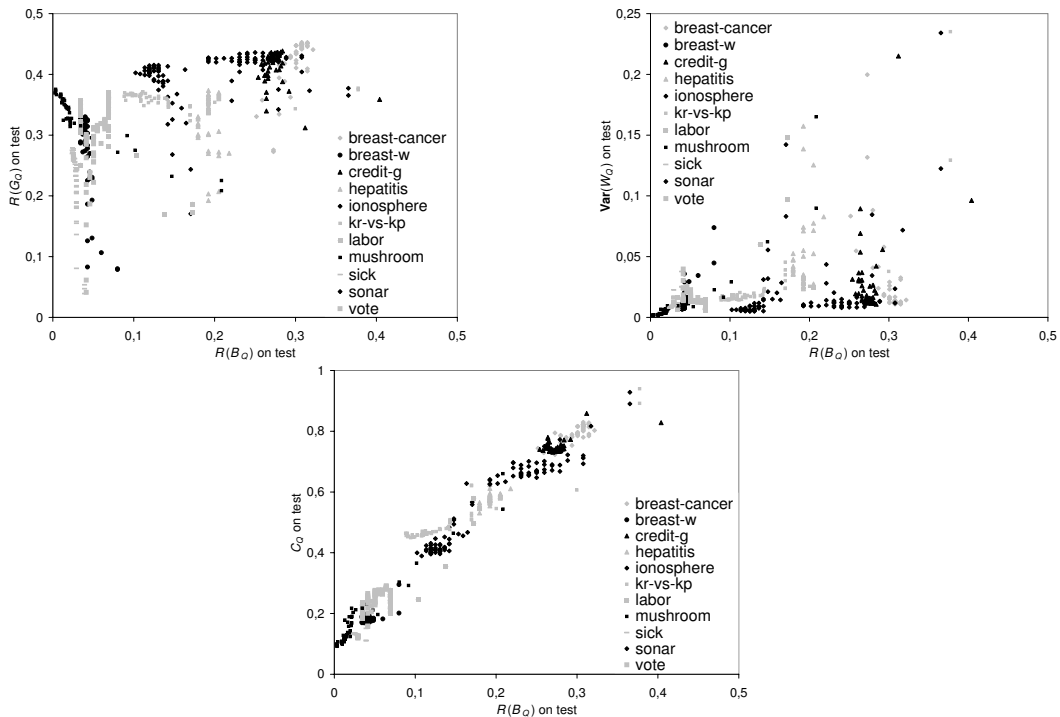

Figure 1: Relation, on various data sets, between $R(B_Q)$ and $R(G_Q)$, $\mathbf{Var}(W_Q)$, and $C_Q$.

## 4   New PAC-Bayes Theorems

A uniform estimate of $C_Q$ can be obtained if we have uniform upper bounds on $R(G_Q)$ and on the variance of $W_Q$. While the original PAC-Bayes theorem provides an upper bound on $R(G_Q)$ that holds uniformly for all posteriors $Q$, obtaining such bounds for the variance of a random variable is still an issue. To achieve this goal, we will have to generalize the PAC-Bayes theorem for expectations over pairs of classifiers since $\mathbf{E}(W_Q^2)$ is fundamentally such an expectation.

**Definition 4**  For any probability distribution $Q$ over $\mathcal{H}$, we define the *expected joint error* ($e_Q$), the *expected joint success* ($s_Q$), and the *expected disagreement* ($d_Q$) as

$$
e_Q \ \stackrel{\text{def}}{=} \ \underset{h_1\sim Q}{\mathbf{E}}\ \underset{h_2\sim Q}{\mathbf{E}}\ \Big(\underset{(\mathbf{x},y)\sim D}{\mathbf{E}}\ I(h_1(\mathbf{x})\neq y)I(h_2(\mathbf{x})\neq y)\Big)
$$

$$
s_Q \ \stackrel{\text{def}}{=} \ \underset{h_1\sim Q}{\mathbf{E}}\ \underset{h_2\sim Q}{\mathbf{E}}\ \Big(\underset{(\mathbf{x},y)\sim D}{\mathbf{E}}\ I(h_1(\mathbf{x})= y)I(h_2(\mathbf{x})= y)\Big)
$$

$$
d_Q \ \stackrel{\text{def}}{=} \ \underset{h_1\sim Q}{\mathbf{E}}\ \underset{h_2\sim Q}{\mathbf{E}}\ \Big(\underset{(\mathbf{x},y)\sim D}{\mathbf{E}}\ I(h_1(\mathbf{x})\neq h_2(\mathbf{x}))\Big).
$$

The empirical estimates, over a training set $S=\langle \mathbf{z}_1,\dots,\mathbf{z}_m\rangle$, of these expectations are defined as usual, i.e., $\widehat{e_Q}\stackrel{\text{def}}{=}\underset{h_1\sim Q}{\mathbf{E}}\underset{h_2\sim Q}{\mathbf{E}}\big(\frac{1}{m}\sum_{i=1}^m I(h_1(\mathbf{x})\neq y)I(h_2(\mathbf{x})\neq y)\big)$, etc.

It is easy to see that

$$
e_Q=\underset{(\mathbf{x},y)\sim D}{\mathbf{E}}W_Q^2, \quad s_Q=\underset{(\mathbf{x},y)\sim D}{\mathbf{E}}(1-W_Q)^2, \quad \text{and} \quad d_Q=\underset{(\mathbf{x},y)\sim D}{\mathbf{E}}2W_Q(1-W_Q). \tag{7}
$$

Thus, we have $e_Q+s_Q+d_Q=1$ and $2e_Q+d_Q=2R(G_Q)$. This implies,

$$
R(G_Q) \ = \ e_Q+\frac{1}{2}\cdot d_Q \ = \ \frac{1}{2}\cdot(1+e_Q-s_Q) \tag{8}
$$

$$
\mathbf{Var}(W_Q) \ = \ e_Q-(R(G_Q))^2 \ = \ e_Q-(e_Q+\frac{1}{2}\cdot d_Q)^2 \ = \ e_Q-\frac{1}{4}\cdot(1+e_Q-s_Q)^2 \tag{9}
$$

Moreover, in that new setting, the denominator of $C_Q$ can elegantly be rewritten as

$$
\mathbf{Var}(W_Q)+(1/2-R(G_Q))^2=1/4-d_Q/2. \tag{10}
$$

The next theorem can be used to bound *separately* either $e_Q$, $s_Q$ or $d_Q$.

**Theorem 5**  *For any prior distribution $P$ over $\mathcal{H}$, and any $\delta\in\,]0,1]$, we have:*

$$
\underset{S\sim D^m}{\Pr}\left(\forall Q\ over\,\mathcal{H}:\ \mathrm{kl}(\widehat{\alpha_Q}\|\alpha_Q)\leq\frac{1}{m}\left[2\cdot\mathrm{KL}(Q\|P)+\ln\frac{(m+1)}{\delta}\right]\right)\geq 1-\delta
$$

*where $\alpha_Q$ can be either $e_Q$, $s_Q$ or $d_Q$.*

In contrast with Theorem 5, the next theorem will enable us to bound directly $\mathbf{Var}(W_Q)$, by bounding any pair of expectations among $e_Q$, $s_Q$ and $d_Q$.

**Theorem 6**  *For any prior distribution $P$ over $\mathcal{H}$, and any $\delta\in\,]0,1]$, we have:*

$$
\underset{S\sim D^m}{\Pr}\left(\forall Q\ over\,\mathcal{H}:\ \mathrm{kl}(\widehat{\alpha_Q},\widehat{\beta_Q}\|\alpha_Q,\beta_Q)\leq\frac{1}{m}\left[2\cdot\mathrm{KL}(Q\|P)+\ln\frac{(m+1)(m+2)}{2\delta}\right]\right)\geq 1-\delta
$$

*where $\alpha_Q$ and $\beta_Q$ can be any two distinct choices among $e_Q$, $s_Q$ and $d_Q$, and where*

$$
\mathrm{kl}(q_1,q_2\|p_1,p_2)\ \stackrel{\text{def}}{=}\ q_1\ln\frac{q_1}{p_1}+q_2\ln\frac{q_2}{p_2}+(1-q_1-q_2)\ln\frac{1-q_1-q_2}{1-p_1-p_2}
$$

*is the Kullback-Leibler divergence between the distributions of two trivalent random variables $Y_q$ and $Y_p$ with $P(Y_q\!=\!a)=q_1$, $P(Y_q\!=\!b)=q_2$ and $P(Y_q\!=\!c)=1-q_1-q_2$ (and similarly for $Y_p$).*

The proof of Theorem 5 can be seen as a special case of Theorem 1. The proof of Theorem 6 essentially follows the proof of Theorem 1 given in [4]; except that it is based on a trinomial distribution instead of a binomial one[2].

# 5 PAC-Bayes Bounds for $\mathbf{Var}(W_Q)$ and $R(B_Q)$

From the two theorems of the preceding section, one can easily derive several PAC-Bayes bounds of the variance of $W_Q$ and therefore, of the majority vote. Since $C_Q$ is a quotient. Thus, an upper bound on $C_Q$ will degrade rapidly if the bounds on the numerator and the denominator are not tight — especially for majority votes obtained by boosting algorithms where both the numerator and the denominator tend to be small. For this reason, we will derive more than one PAC-Bayes bound for the majority vote, and compare their accuracy. First, we need the following notations that are related to Theorems 1, 5 and 6. Given any prior distribution $P$ over $\mathcal{H}$,

$$\mathcal{R}_{Q,S}^{\delta} \;\overset{\text{def}}{=}\; \left\{ r \;:\; \mathrm{kl}(R_S(G_Q)\|r) \leq \frac{1}{m}\left[\mathrm{KL}(Q\|P) + \ln\frac{(m+1)}{\delta}\right]\right\},$$

$$\mathcal{E}_{Q,S}^{\delta} \;\overset{\text{def}}{=}\; \left\{ e \;:\; \mathrm{kl}(\widehat{e_Q}\|e) \leq \frac{1}{m}\left[2{\cdot}\mathrm{KL}(Q\|P) + \ln\frac{(m+1)}{\delta}\right]\right\},$$

$$\mathcal{D}_{Q,S}^{\delta} \;\overset{\text{def}}{=}\; \left\{ d \;:\; \mathrm{kl}(\widehat{d_Q}\|d) \leq \frac{1}{m}\left[2{\cdot}\mathrm{KL}(Q\|P) + \ln\frac{(m+1)}{\delta}\right]\right\},$$

$$\mathcal{A}_{Q,S}^{\delta} \;\overset{\text{def}}{=}\; \left\{ (e,s) \;:\; \mathrm{kl}(\widehat{e_Q},\widehat{s_Q}\|e,s) \leq \frac{1}{m}\left[2{\cdot}\mathrm{KL}(Q\|P) + \ln\frac{(m+1)(m+2)}{\delta}\right]\right\}.$$

Since $\frac{v}{v+a} = \frac{1}{1+a/v}$, it follows from Theorem 2 that an upper bound of both $\mathbf{Var}(W_Q)$ and $R(G_Q)$ will give an upper bound on $C_Q$, and hence on $R(B_Q)$. Hence, a first bound can be obtained, from Equation 9, by suitably applying Theorem 5 (with $\alpha_Q = e_Q$) and Theorem 1.

**PAC-Bound 1** *For any prior distribution $P$ over $\mathcal{H}$, and any $\delta \in\,]0,1]$, we have*

$$\Pr_{S\sim D^m}\left(\forall Q \text{ over } \mathcal{H}: \quad \mathbf{Var}_{(\mathbf{x},y)\sim D} W_Q \;\leq\; \sup \mathcal{E}_{Q,S}^{\delta/2} - \left(\inf \mathcal{R}_{Q,S}^{\delta/2}\right)^2\right) \geq 1-\delta\,,$$

$$\Pr_{S\sim D^m}\left(\forall Q \text{ over } \mathcal{H}: \quad R(B_Q) \;\leq\; \frac{\sup \mathcal{E}_{Q,S}^{\delta/2} - \left(\inf \mathcal{R}_{Q,S}^{\delta/2}\right)^2}{\sup \mathcal{E}_{Q,S}^{\delta/2} - \left(\inf \mathcal{R}_{Q,S}^{\delta/2}\right)^2 + \left(\frac{1}{2} - \sup \mathcal{R}_{Q,S}^{\delta/2}\right)^2}\right) \geq 1-\delta\,.$$

Since Bound 1 necessitates two PAC approximations to calculate the variance, it would be better if we could obtain directly an upper bound for $\mathbf{Var}(W_Q)$. The following result, which is a direct consequence of Theorem 6 and Equation 9, shows how it can be done.

**PAC-Bound 2** *For any prior distribution $P$ over $\mathcal{H}$, and any $\delta \in\,]0,1]$, we have*

$$\Pr_{S\sim D^m}\left(\forall Q \text{ over } \mathcal{H}: \quad \mathbf{Var}_{(\mathbf{x},y)\sim D} W_Q \;\leq\; \sup_{(e,s)\in\mathcal{A}_{Q,S}^{\delta}}\left\{e - \frac{1}{4}\cdot(1+e-s)^2\right\}\right) \geq 1-\delta\,,$$

$$\Pr_{S\sim D^m}\left(\forall Q \text{ over } \mathcal{H}: \quad R(B_Q) \leq \frac{\displaystyle\sup_{(e,s)\in\mathcal{A}_{Q,S}^{\delta/2}}\left\{e - \tfrac{1}{4}\cdot(1+e-s)^2\right\}}{\displaystyle\sup_{(e,s)\in\mathcal{A}_{Q,S}^{\delta/2}}\left\{e - \tfrac{1}{4}\cdot(1+e-s)^2\right\} + \left(\tfrac{1}{2} - \sup \mathcal{R}_{Q,S}^{\delta/2}\right)^2}\right) \geq 1-\delta\,.$$

As illustrated in Figure 2, Bound 2 is generally tighter than Bound 1. This gain is principally due to the fact that the values of $e$ and $s$, that are used to bound the variance, are tied together inside the $\mathrm{kl}()$ and have to tradeoff their values ($e$ "tries to be" as large as possible and $s$ as small as possible). Because of this tradeoff, $e$ is generally not an upper bound of $e_Q$, and $s$ not a lower bound of $s_Q$.

In the semi-supervised framework, we can achieve better results, because the labels of the examples do not affect the value of $d_Q$ (see Definition 4). Hence, in presence of a large amount of unlabelled data, one can use Theorem 5 to obtain very accurate upper and lower bounds of $d_Q$. This combined with an upper bound of $e_Q$, still computed via Theorem 5 but on the labelled data, gives rise to the

following semi-supervised upper bound[3] of $\mathbf{Var}(W_Q)$. The bound on $R(B_Q)$ then follows from Theorem 2 and Equation 10.

**PAC-Bound 3** *(semi-supervised bound) For any prior distribution $P$ over $\mathcal{H}$, and any $\delta \in \,]0, 1]$:*

$$\Pr_{\substack{S \sim D^m \\ S' \sim D_{unlabelled}^{m'}}} \left( \forall Q \; over \, \mathcal{H} : \; \mathbf{Var}_{(\mathbf{x},y) \sim D} W_Q \; \leq \; \sup \mathcal{E}_{Q,S}^{\delta} - \left( \sup \mathcal{E}_{Q,S}^{\delta} + \frac{1}{2} \cdot \inf \mathcal{D}_{Q,S'}^{\delta} \right)^2 \right) \geq 1 - \delta$$

$$\Pr_{\substack{S \sim D^m \\ S' \sim D_{unlabelled}^{m'}}} \left( \forall Q \; over \, \mathcal{H} : \quad R(B_Q) \; \leq \; \frac{\sup \mathcal{E}_{Q,S}^{\delta} - \left( \sup \mathcal{E}_{Q,S}^{\delta} + \frac{1}{2} \cdot \inf \mathcal{D}_{Q,S'}^{\delta} \right)^2}{1/4 - \frac{1}{2} \cdot \sup \mathcal{D}_{Q,S'}^{\delta}} \right) \geq 1 - \delta$$

We see, on the left part of Figure 2, that Bound 2 on $\mathbf{Var}(W_Q)$ is much tighter than Bound 1. We can also see that, by using unlabeled data[4] to estimate $d_Q$, Bound 3 provides an other significant improvement. These numerical results were obtained by using Adaboost [9] with decision stumps on the Mushroom UCI data set (which contains 8124 examples). This data set was randomly split into two halves: one for training and one for testing.

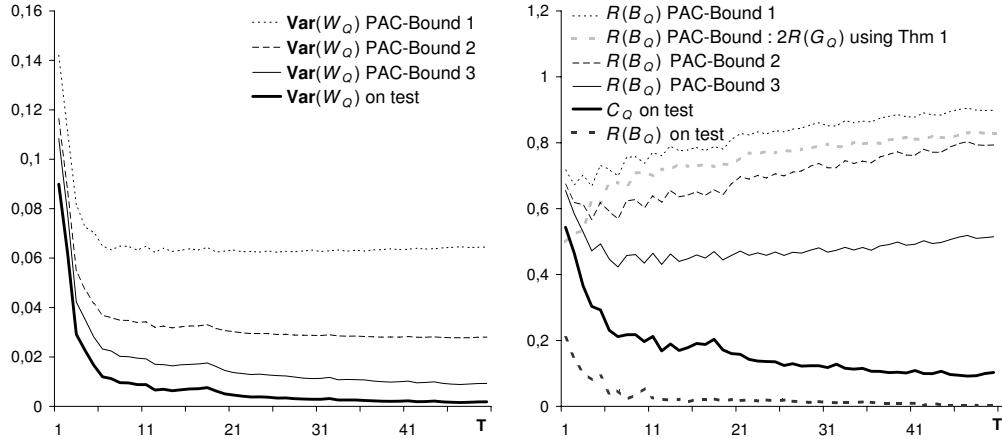

Figure 2: Bounds on $\mathbf{Var}(W_Q)$ (left) and bounds on $R(B_Q)$ (right).

As illustrated by Figure 2, Bound 2 and Bound 3 are (resp. for the supervised and semi-supervised frameworks) very tight upper bounds of the variance. Unfortunately they do not lead to tight upper bounds of $R(B_Q)$. Indeed, one can see in Figure 2 that after $T = 8$, all the bounds are degrading even if the true value of $C_Q$ (on which they are based) continues to decrease. This drawback is due to the fact that, when the value of $d_Q$ tends to $1/2$, the denominator of $C_Q$ tends to $0$. Hence, if $d_Q$ is close to $1/2$, $\mathbf{Var}(W_Q)$ must be small as well. Thus, any slack in the bound of $\mathbf{Var}(W_Q)$ has a multiplicative effect on each of the three proposed PAC-bounds of $R(B_Q)$. Unfortunately, boosting algorithms tend to construct majority votes with expected an disagreement $d_Q$ just slightly under $1/2$. Based on the next proposition, we will show that this drawback is, in a sense, unavoidable.

**Proposition 7** *(Inapproachability result) Let $Q$ be any distribution over a class of classifiers, and let $\overline{B} < 1$ be any upper bound of $C_Q$ which holds with confidence $1 - \delta$. If $R(G_Q) < 1/2$ then*

$$1/2 - \sqrt{(1/4 - d_Q/2)\left(1 - \overline{B}\right)}$$

*is an upper bound of $R(G_Q)$ which holds with confidence $1 - \delta$.*

For the data set used in Figure 2, Proposition 7, together with Bound 3 on $R(B_Q)$ (viewed as a bound on $C_Q$), gives a PAC-bound on $R(G_Q)$ which is just slightly lower ($\approx 0.5\%$) than the classical PAC-Bayes bound on $R(G_Q)$ given by Theorem 1. Since any bound better than Bound 3 for $C_Q$ will continue to improve the bound on $R(G_Q)$, it seems unlikely that such a better bound exists. Moreover, this drawback should occur for any bound on the majority vote that only considers Gibbs' risk and the variance of $W_Q$ because, as already explained, $C_Q$ is the tightest possible bound of $R(B_Q)$ that is based only on $\mathbf{E}(W_Q)$ and $\mathbf{Var}(W_Q)$. Hence, to improve our results in the situation where $d_Q$ is closed to $1/2$, one will have to consider higher moments. However, it is not clear that this will lead to a better bound of $R(B_Q)$ because, even if Theorem 5 generalizes to higher moments, its tightness is then degrading. Indeed, for the $k^{\text{th}}$ moment, the factor 2 that multiplies $\mathrm{KL}(Q\|P)$ in Theorem 5 grows to $k$. However, it might be possible to overcome this degradation by using a generalization of Theorem 6 as we have done in this paper to obtain our tightest supervised bound for the variance (Bound 2). Indeed, if we evaluate the tightness of that bound on the variance (w.r.t. its value on the test set), and compare it with the tightness of the bound on $R(G_Q)$ given by Theorem 1, we find that both accuracies are at about $3\%$. This is to be contrasted with the tightness of Bound 1 and seems to indicate that we have prevented degradation even if the variance deals with both the first and the second moment of $W_Q$; whereas the Gibbs' risk deals only with the first moment.

## 6   Conclusion

We have derived a risk bound for the weighted majority vote that depends on the mean and variance of the error of its associated Gibbs classifier (Theorem 2). The proposed bound is based on the one-sided Chebychev's inequality, which is the tightest inequality for any real-valued random variables given only the expectation and the variance. As shown on Figures 1, this bound seems to have a strong predictive power on the risk of the majority vote.

We have also shown that the original PAC-Bayes Theorem, together with new ones, can be used to obtain high confidence estimates of this new risk bound that hold uniformly for *all* posterior distributions. Moreover, the new PAC-Bayes theorems give rise to the first uniform bounds on the variance of the Gibbs's risk (more precisely, the variance of the associate random variable $W_Q$). Even if there are arguments showing that bounds of higher moments of $W_Q$ should be looser, we have empirically found that one of the proposed bounds (Bound 2) does not show any sign of degradation in comparison with the classical PAC-Bayes bound on $R(G_Q)$ (which is the first moment). Surprisingly, there is an *improvement* for Bound 3 in the semi-supervised framework. This also opens up the possibility that the generalization of Theorem 2 to higher moment be applicable to real data. Such generalizations might overcome the main drawback of our approach, namely, the fact that the PAC-bounds, based on Theorem 2, degrade when the expected disagreement ($d_Q$) is close to $1/2$.

**Acknowledgments:**  Work supported by NSERC Discovery grants 262067 and 0122405.

## Footnotes

[1]Priors have been used for many years in statistics. The priors in this paper have only indirect links with the *Bayesian priors*. We will nevertheless use this language, since it comes from previous work.

[2]For the proofs of these theorems, see a long version of the paper at `http://www.ift.ulaval.ca/~laviolette/Publications/publications.html`.

[3]It follows, from an easy calculation, that a lower bound of $d_Q$, together with an upper bound of $e_Q$, gives rise to an upper bound of $e_Q - (e_Q + \frac{1}{2} \cdot d_Q)^2$. By Equation 9, we then obtain an upper bound of $\mathbf{Var}(W_Q)$.

[4]The UCI database (used here) does not have any unlabeled examples. To simulate the extreme case where we have an infinite amount of unlabeled data, we simply used the empirical value of $d_Q$ computed on the testing set.

## References

[1] David McAllester. Some PAC-Bayesian theorems. *Machine Learning*, 37:355–363, 1999.

[2] David McAllester. PAC-Bayesian stochastic model selection. *Machine Learning*, 51:5–21, 2003.

[3] David McAllester. Simplified PAC-Bayesian margin bounds. *Proceedings of the 16th Annual Conference on Learning Theory, Lecture Notes in Artificial Intelligence*, 2777:203–215, 2003.

[4] François Laviolette and Mario Marchand. PAC-Bayes risk bounds for sample-compressed Gibbs classifiers. *Proc. of the 22nth International Conference on Machine Learning (ICML 2005)*, pages 481–488, 2005.

[5] John Langford and John Shawe-Taylor. PAC-Bayes & margins. In S. Thrun S. Becker and K. Obermayer, editors, *Advances in Neural Information Processing Systems 15*, pages 423–430. MIT Press, Cambridge, MA, 2003.

[6] Luc Devroye, László Györfi, and Gábor Lugosi. *A Probabilistic Theory of Pattern Recognition*. Springer Verlag, New York, NY, 1996.

[7] Leo Breiman. Random forests. *Machine Learning*, 45(1):5–32, 2001.

[8] Dimitris Bertsimas and Ioana Popescu. Optimal inequalities in probability theory: A convex optimization approach. *SIAM J. on Optimization*, 15(3):780–804, 2005.

[9] Robert E. Schapire and Yoram Singer. Improved boosting using confidence-rated predictions. *Machine Learning*, 37(3):297–336, 1999.